# Randomized Pruning: Efficiently Calculating Expectations in Large Dynamic Programs

**Alexandre Bouchard-Côté**[1]
bouchard@cs.berkeley.edu

**Slav Petrov**[2,†]
slav@google.com

**Dan Klein**[1]
klein@cs.berkeley.edu

[1]Computer Science Division
University of California at Berkeley
Berkeley, CA 94720

[2]Google Research
76 Ninth Ave
New York, NY 10011

## Abstract

Pruning can massively accelerate the computation of feature expectations in large models. However, any single pruning mask will introduce bias. We present a novel approach which employs a randomized sequence of pruning masks. Formally, we apply auxiliary variable MCMC sampling to generate this sequence of masks, thereby gaining theoretical guarantees about convergence. Because each mask is generally able to skip large portions of an underlying dynamic program, our approach is particularly compelling for high-degree algorithms. Empirically, we demonstrate our method on bilingual parsing, showing decreasing bias as more masks are incorporated, and outperforming fixed tic-tac-toe pruning.

## 1 Introduction

Many natural language processing applications, from discriminative training [18, 9] to minimum-risk decoding [16, 34], require the computation of expectations over large-scale combinatorial spaces. Problem scale comes from a combination of large constant factors (such as the massive grammar sizes in monolingual parsing) or high-degree algorithms (such as the many dimensions of bitext parsing). In both cases, the primary mechanism for efficiency has been pruning, wherein large regions of the search space are skipped on the basis of some computation mask. For example, in monolingual parsing, entire labeled spans may be skipped on the basis of posterior probabilities in a coarse grammar [17, 7]. Conditioning on these masks, the underlying dynamic program can be made to run arbitrarily quickly.

Unfortunately, aggressive pruning introduces biases in the resulting expectations. As an extreme example, features with low expectation may be pruned down to zero if their supporting structures are completely skipped. One option is to simply prune less aggressively and spend more time on a single, more exhaustive expectation computation, perhaps by carefully tuning various thresholds [26, 12] and using parallel computing [9, 38]. However, we present a novel alternative: randomized pruning. In randomized pruning, *multiple* pruning masks are used in sequence. The resulting sequence of expectation computations are averaged, and errors average out over the multiple computations. As a result, time can be directly traded against approximation quality, and errors of any single mask can be overcome.

Our approach is based on the idea of *auxiliary variable sampling* [31], where a set of auxiliary variables formalizes the idea of a pruning mask. Resampling the auxiliary variables changes the mask at each iteration, so that the portion of the chart that is unconstrained at a given iteration can improve the mask for subsequent iterations. In other words, pruning decisions are continuously revisited and revised. Since our approach is formally grounded in the framework of block Gibbs sampling [33], it inherits desirable guarantees as a consequence. If one needs successively better

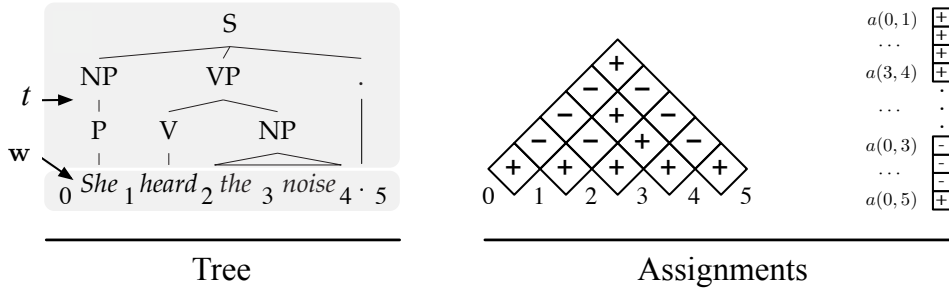

| Tree | Assignments |
|---|---|

Figure 1: A parse tree, from which the assignment variables are extracted. A linearization into an assignment vector is shown at the right.

approximations, more iterations can be performed, with a guarantee of convergence to the true expectations.

In practice, of course, we are only interested in the behavior after a finite number of iterations: the method would be useless if it did not outperform previous heuristics in a time range bounded by the exact computation time. Here, we investigate empirical performance on English-Chinese bitext parsing, showing that bias decreases over time. Moreover, we show that our randomized pruning outperforms standard single-mask tic-tac-toe pruning [40], achieving lower bias over a range of total computation times. Our technique is orthogonal to approaches that use parallel computation [9, 38], and can be additionally parallelized at the sentence level.

In what follows, we explain the method in the context of parsing because it makes the exposition more concrete, and because our experiments are on similar combinatorial objects (bitext derivations). Note, however, that the applicability of this approach is in no way limited to parsing. The settings in which randomized pruning will be most advantageous will be those in which high-order dynamic programs can be vastly sped up by masking, yet no single aggressive mask is likely to be adequate.

## 2 Randomized pruning

### 2.1 The need for expectations

Algorithms for discriminative training, consensus decoding, and unsupervised learning typically involve repetitively computing a large number of expectations. In discriminative training of probabilistic parsers, for example [18, 32], one needs to repeatedly parse the entire training set in order to compute the necessary expected feature counts. In this setup (Figure 1), the conditional distribution of a tree-valued random variable $T$ given a yield $\mathbf{y}(T) = w$ is modeled using a log-linear model : $\mathbb{P}_\theta(T = t | \mathbf{y}(T) = w) = \exp\{\langle \theta, f(t, w) \rangle - \log Z(\theta, w)\}$, in which $\theta \in \mathbb{R}^K$ is a parameter vector and $f(t, w) \in \mathbb{R}^K$ is a feature function. Training such a model involves the computation of the following gradient in between each update of $\theta$:

$$\nabla \log \prod_{i \in \mathcal{I}} \mathbb{P}_\theta(T = t_i | \mathbf{y}(T) = w_i) = \sum_{i \in \mathcal{I}} \left\{ f(t_i, w_i) - \mathbb{E}_\theta[f(T, w_i) | \mathbf{y}(T) = w_i] \right\},$$

where $\{w_i : i \in \mathcal{I}\}$ are the training sentences with corresponding gold trees $\{t_i\}$.

The first term in the above equation can be computed in linear time, while the second requires a cubic-time dynamic program (the inside-outside algorithm), which computes constituent posteriors for all possible spans of words (the chart cells in Figure 1). Hence, computing expectations is indeed the bottleneck here. While it is not impossible to calculate these expectations exactly, it is computationally very expensive, limiting previous work to toy setups with 15 word sentences [18, 32, 35], or necessitating aggressive pruning [26, 12] that is not well understood.

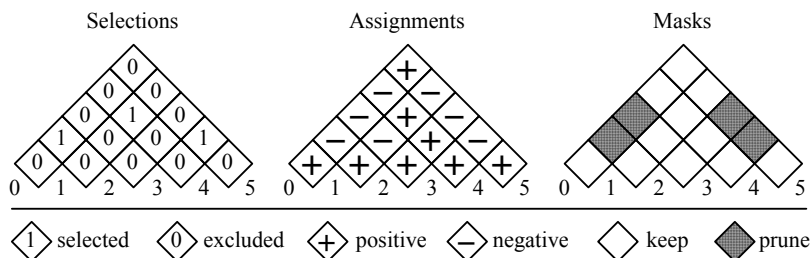

Figure 2: An example of how a selection vector $s$ and an assignment vector $a$ are turned into a pruning mask $m$.

## 2.2 Approximate expectations with a single pruning mask

In the case of monolingual parsing, the computation of feature count expectations is usually approximated with a *pruning mask*, which allows the omission of low probability constituents. Formally, a pruning mask is a map from the set $M$ of all possible spans to the set {prune, keep}, indicating whether a given span is to be ignored. It is easy to incorporate such a pruning mask into existing dynamic programming algorithms for computing expectations: Whenever a dynamic programming state is considered, we first consult the mask and skip over the pruned states, greatly accelerating the computation (see Algorithm 3 for a schematic description of the pruned inside pass). However, the expected feature counts $\mathbb{E}_m[f]$ computed by pruned inside-outside with a single mask $m$ are not exact, introducing a systematic error and biasing the model in undesirable ways.

## 2.3 Approximate expectations with a sequence of masks

To reduce the bias resulting from the use of a single pruning mask, we propose a novel algorithm that can combine several masks. Given a sequence of masks, $m^{(1)}, m^{(2)}, \ldots, m^{(N)}$, we will average the expectations under each of them $\frac{1}{N} \sum_{i=1}^{N} \mathbb{E}_{m^{(i)}}[f]$. Our contribution is to show a principled way of computing a sequence of masks such that this average not only has theoretical guarantees, but also has good finite-sample performance. The key is to define a set of auxiliary variables, and we present this construction in more detail in the following sections. In this section, we present the algorithm operationally.

The masks are defined via two vector-valued Markov chains: a *selection* chain with current value denoted by $s$, and an *assignment* chain with current value $a$. Both $s$ and $a$ are vectors with coordinates indexed by spans over the current sentence: $\iota \in M = \{\langle j, k \rangle : 0 \leq j < k \leq n = |w|\}$. Elements $s_\iota$ specify whether a span $\iota$ will be *selected* ($s_\iota = 1$) or *excluded* (0) in the current iteration $(i)$. The *assignment* vector $a$ then determines, for each span, whether it would be forbidden if selected (or *negative*, $a_\iota = -$) or required (*positive*, $+$) to be a constituent.

Our masks $m = m(s, a)$ are generated deterministically from the selection and assignment vectors. The deterministic procedure uses $s$ to pick a few spans and values to fix from $a$, forming a mask $m$. Note that a single span $\iota$ that is both positive and selected implies that all of the spans $\kappa$ crossing $\iota$ should be pruned (i.e. all of the spans such that neither $\iota \subseteq \kappa$ nor $\kappa \subseteq \iota$ holds). This compilation of the pruning constraints is described in Algorithm 2. The type of the return value $m$ of this function is also a vector with coordinates corresponding to spans: $m_\iota \in$ {prune, keep}. Computation of this mask is illustrated on a concrete example in Figure 2.[1]

We can now summarize how randomized pruning works (see Algorithm 1 for pseudocode). At the beginning of every iteration $(i)$, the first step is to sample new values of the selection vector conditioning on the current selection vector. We will refer to the transition probability of this Markov chain on selection vectors as $k^*$. Once a new mask $m$ has been precomputed from the current selection vector and assignments, pruned inside-outside scores are computed using this mask. The

| **Algorithm 1** : **AuxVar**(w, $f$) | **Algorithm 2** : **CreateMask**($s$,$a$) | **Algorithm 3** : **PrunedInside**(w, $m$) |
|---|---|---|
| $a, s \leftarrow$ random initialization | **for** $\iota \in M$ **do** | {Initialize the chart in the standard way} |
| $E \leftarrow 0$ |   **for** $\kappa \in s$ **do** | **for** $\iota = \langle j, k \rangle \in M$, bottom-up **do** |
| **for** $i \in 1, 2, \ldots, N$ **do** |     **if** $a_\iota = -$ **and** $\iota = \kappa$ **then** |   **if** $m_\iota =$ keep **then** |
|   $s \sim k^*(s, \cdot)$ |       $m_\iota \leftarrow$ prune |     **for** $l : j < l < k$ **do** |
|   $m \leftarrow$ **CreateMask**($s, a$) |       continue outer loop |       **if** $m_{\langle j,l \rangle} = m_{\langle l,k \rangle} =$ keep **then** |
|   Compute **PrunedInside**(w,$m$) |     **if** $a_\iota = +$ **and** $\iota \not\subseteq \kappa$ |         {Loop over grammar symbols, |
|   Compute **PrunedOutside**(w,$m$) |     **and** $\kappa \not\subseteq \iota$ **then** |         update inside scores in the |
|   $E \leftarrow E + \mathbb{E}_m f$ |       $m_\iota \leftarrow$ prune |         standard way} |
|   $a \sim k_s(a, \cdot)$ |       continue outer loop | |
| **return** $\frac{E}{N}$ |   $m_\iota \leftarrow$ keep | **return** chart |
| | **return** $m$ | |

Figure 3: Pseudo-code for randomized pruning in the case of monolingual parsing (assuming a grammar with no unaries except at pre-terminal positions. We have omitted **PrunedOutside** because of limited space, but its structure is very similar to **PrunedInside**.

inside-outside scores are then used in two ways: first, to calculate the expected feature counts under the pruned model, $\mathbb{E}_m[f]$, which are added to a running average; second, to resample new values for the assignment vector.[2]

Let us describe in more detail how a new assignment vector $a'$ is updated given the previous assignment $a$. This is a two step update process. First, a tree $t$ is sampled from the chart computed by **PrunedInside**$(w, m)$ (Figure 1, left). This can be done in quadratic time using a standard algorithm [19, 13]. Next, the assignments are set to a new value deterministically as follows: for each span $\iota$, $a_\iota = +$ if $\iota$ is a constituent in $t$, and $a_\iota = -$ otherwise (Figure 1, right). We will denote this property by $[\iota \in t]$.

We defer to Section 3.2 for the description of the selection vector updates—the form of these updates will be easier to motivate after the analysis of the algorithm.

## 3 Analysis

In this section we show that the procedure described above can be viewed as running an MCMC algorithm. This implies that the guarantees associated with this class of algorithms extend to our procedure. In particular, consistency holds: $\frac{1}{N} \sum_{i=1}^{N} \mathbb{E}_{m^{(i)}} f \xrightarrow{a.s.} \mathbb{E} f$.

### 3.1 Auxiliary variables and the assignment Markov chain

We start by formally describing the Markov chain over assignments. This is done by defining a collection of Gibbs operators $k_s(\cdot, \cdot)$ indexed by a selection vectors $s$.

The original state space (the space of trees) does not easily decompose into a graphical model where textbook Gibbs sampling could be applied, so we first *augment* the state space with auxiliary variables. Broadly speaking, an auxiliary variable is a state augmentation such that the target distribution is a marginal of the expanded distribution. It is called auxiliary because the parts of the samples corresponding to the augmentation are discarded at the end of the computation. At an intermediate stage, however, the state augmentation helps explore the space efficiently.

This technique is best explained with a concrete example in our parsing setup. In this case, the augmentation is a collection of $|M|$ binary-valued random variables, each corresponding to a span of the current sentence $w$. The auxiliary variable corresponding to span $\iota \in M$ will be denoted by $A_\iota$. We define the auxiliary variables by specifying their conditional distribution $A_\iota | (T = t)$. This conditional is a deterministic function: $\mathbb{P}(A_\iota | T = t) = [\iota \in t]$.

With this augmentation, we can now describe the sampler. It is a *block* Gibbs sampler, meaning that it resamples a subset of the random variables, conditioning on the other ones. Even when the subsets selected across iterations overlap, acceptance probabilities are still guaranteed to be one [33].

The blocks of resampled variables will always contain $T$ as well as a subset of the *excluded* auxiliary variables. Note that when conditioning on all of the auxiliary variables, the posterior distribution on $T$ is deterministic. We therefore require that $\mathbb{P}(|s| < |M| \text{ i.o.}) = 1$ to maintain irreducibility.

We now describe in more detail the effect that each setting of $a, s$ has on the posterior distribution on $T$. We start by developing the form of the posterior distribution over trees when there is a single selected auxiliary variable, i.e. $T|(A_\iota = a)$. If $a = -$, sampling from $T|A_\iota = \iota$ requires the same dynamic program as for exact sampling, except that a single cell in the chart is pruned (the cell $\iota$). The setting where $a = +$ is more interesting: in this case significantly more cells can be pruned. Indeed, all constituents overlapping with $\iota$ are pruned. This can lead to a speed-up of up to a multiplicative constant of $8 = 2^3$, when the span $\iota$ has length $|\iota| = \frac{|w|}{2}$. More constraints are maintained during resampling steps in practice (i.e. $|s| > 1$), leading to a large empirical speedup.

Consider now the problem of jointly resampling the block containing $T$ and a collection of excluded auxiliary variables $\{A_\iota : \iota \notin s\}$ given a collection of selected ones. We can write the decomposition:

$$\mathbb{P}(T = t, \mathcal{S}|\mathcal{C}) = \mathbb{P}(T = t|\mathcal{C}) \prod_{\iota \notin s} \mathbb{P}(A_\iota = a_\iota|T = t)$$

$$= \mathbb{P}(T = t|\mathcal{C}) \prod_{\iota \notin s} \mathbf{1}\{a_\iota = [\iota \in t]\},$$

where $\mathcal{S} = \big(A_\iota = a_\iota : \iota \notin s\big)$ is a configuration of the excluded auxiliary variables and $\mathcal{C} = \big(A_\iota = a_\iota : \iota \in s\big)$ is a configuration of the selected ones. The first factor in the second line is again a pruned dynamic program (described in Algorithm 3). The product of indicator functions shows that once a tree has been picked, the excluded auxiliary variables can be set to new values deterministically by reading from the sampled tree $t$ whether $\iota$ is a constituent, for each $\iota \notin s$.

Given a selection vector $s$, we denote the induced block Gibbs kernel described above by $k_s(\cdot, \cdot)$. Since this kernel depends on the previous state only through the assignments of the auxiliary variables, we can also write it as a transition kernel on the space $\{+, -\}^{|M|}$ of auxiliary variable assignments: $k_s(a, a')$.

### 3.2   The selection chain

There is a separate mechanism, $k^*$, that updates at each iteration the selection $s$ of the auxiliary variables. This mechanism corresponds to picking which Gibbs operator $k_s$ will be used to transition in the Markov chain on assignments described above. We will denote the random variable corresponding to the selection vector $s$ at state $(i)$ by $S^{(i)}$.

In standard treatments of MCMC algorithms [33, 22], the variables $S^{(i)}$ are restricted to be either independent (a mixture of kernels), or deterministic enumerations (an alternation of kernels). However this restriction can be relaxed to having $S^{(i)}$ be itself a Markov chain with kernel $k^* : \{0, 1\}^{|M|} \times \{0, 1\}^{|M|} \to [0, 1]$. This relaxation can be thought of as allowing stochastic policies for kernel selection.[3]

The choice of $k^*$ is important. To understand why, recall that in the situation where $(A_\iota = -)$, a single cell in the chart is pruned, whereas in the case where $(A_\iota = +)$, a large fraction of the chart can be ignored. The construction of $k^*$ is therefore where having a simpler model or heuristic at hand can play a role: as a way to favor the selection of constituents that are likely to be positive, so that better speedup can be achieved. Note that the algorithm can recover from mistakes in the simpler model, since the assignments of the auxiliary variables are also resampled.

Another issue that should be considered when designing $k^*$ is that it should avoid self-transitions (repeating the same set of selections). To see why, note that if $(s, a) = (s', a')$, then $m = m(s, a) =$

$m(s', a') = m'$ and hence $\frac{\mathbb{E}_m f + \mathbb{E}_{m'} f}{2} = \mathbb{E}_m f$. The estimator is unchanged in this case, even after paying the computational cost of a second iteration.

The mechanism we used takes both of these issues into consideration. First, it uses a simpler model (for instance a grammar with fewer non-terminal symbols) to pick a subset $M' \subseteq M$ of the spans that have high posterior probability. Our kernel $k^*$ is restricted to selection vectors $s$ such that $s \subseteq M'$. Next, in order to avoid repetition, our kernel transitions from a previous selection $s$ to the next one, $s'$, as follows: after picking a random subset $R \subset s$ of size $\frac{|s|}{2}$, define $s' = (M' \backslash s) \cup R$. Provided that the chain is initialized with $|s| = \frac{2|M'|}{3}$, this scheme has the property that it changes a large portion of the state at every iteration (more precisely, $|s \cap s'| = \frac{1}{3}$), and moreover all subsets of $M'$ of size $\frac{2|M'|}{3}$ are eventually resampled with probability one. Note that this update depends on the previous selection vector, but not on the assignment vector.

Given the asymmetric effect between conditioning on positive versus negative auxiliary variables, it is tempting to let the $k^*$ depend on the current assignment of the auxiliary variables. Unfortunately such schemes will not converge to the correct distribution in general. Counterexamples are given in the adaptive MCMC literature [2].

## 3.3 Accelerated averaging

In this section, we justify the way expected sufficient statistics are estimated from the collection of samples. In other words, how the variable $E$ is updated in Algorithm 1.

In a generic MCMC situation, once samples $X^{(1)}, X^{(2)}, \ldots$ are collected, the traditional way of estimating expected sufficient statistics $f$ is to average "hard counts," i.e. to use the estimator: $S_N = \frac{1}{N} \sum_{i=1}^{N} f(X^{(i)})$. In our case $X^{(i)}$ contains the current tree and assignments, $(T^{(i)}, A^{(i)})$.

For general Metropolis-Hastings chains, this is often the only method available. On the other hand, in our parsing setup—and more generally, with any Gibbs sampler—it turns out that there is a more efficient way of combining the samples [23]. The idea behind this alternative is to take "soft counts." This is what we do when we add $\mathbb{E}_m f$ to the running average in Algorithm 1.

Suppose we have extracted samples $X^{(1)}, X^{(2)}, \ldots, X^{(i)}$, with corresponding selection vectors $S^{(1)}, S^{(2)}, \ldots, S^{(i)}$. In order to transition to the next step, we will have to sample from the probability distribution denoted by $k_{S^{(i)}}(X^{(i)}, \cdot)$. In the standard setting, we would extract a single sample $X^{(i+1)}$ and add $f(X^{(i+1)})$ to a running average.

More formally, the accelerated averaging method consists of adding the following soft count instead: $\int f(x) k_{S^{(i)}}(X^{(i)}, \mathrm{d}x)$, which can be computed with one extra pruned outside computation in our parsing setup. This quantity was denoted $\mathbb{E}_m f$ in the previous section. The final estimator then has the form:[4] $S'_N = \frac{1}{N-1} \sum_{i=1}^{N-1} \int f(x) \; k_{S^{(i)}}(X^{(i)}, \mathrm{d}x)$.

## 4 Experiments

While we used the task of monolingual parsing to illustrate our randomized pruning procedure, the technique is most powerful when the dynamic program is a higher-order polynomial. We therefore demonstrate the utility of randomized pruning on a bitext parsing task. In bitext parsing, we have sentence-aligned corpora from two languages, and are computing expectations over aligned parse trees [6, 28]. The model we use is most similar to [3], but we extend this model and allow rules to mix terminals and non-terminals, as is often done in the context of machine translation [8]. These rules were excluded in [3] for tractability reasons, but our sampler allows efficient sampling in this more challenging setup.

In the terminology of adaptor grammars [19], our sampling step involves resampling an adapted derivation given a base measure derivation for each sentence. Concretely, the problem is to sample from a class of *isotonic* bipartite graphs over the nodes of two trees. By isotonic we mean that the

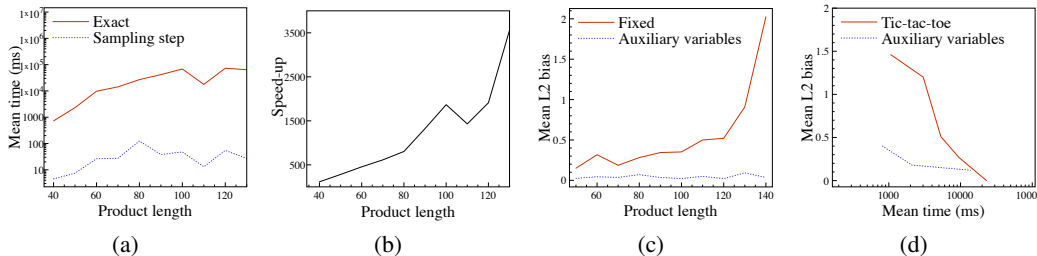

Figure 4: Because each sampling step is three orders of magnitude faster than the exact computation (a,b), we can afford to average over multiple samples and thereby reduce the $\mathbb{L}_2$ bias compared to a fixed pruning scheme (c). Our auxiliary variable sampling scheme also substantially outperforms the tic-tac-toe pruning heuristic (d).

edges $E$ of this bipartite graph should have the property that if two non-terminals $\alpha, \alpha'$ and $\beta, \beta'$ are aligned in the sampled bipartite graph, i.e. $(\alpha, \alpha') \in E$ and $(\beta, \beta') \in E$, then $\alpha \geq \beta \Rightarrow \alpha' \geq \beta'$, where $\alpha \geq \beta$ denotes that $\alpha$ is an ancestor of $\beta$. The weight (up to a proportionality constant) of each of these alignments is obtained as follows: first, consider each aligned point as the left-hand of a rule. Next, multiply the score of these rules. If we let $p, q$ be the length of the two sentences, one can check that this yields a dynamic program of complexity $O(p^{b+1}q^{b+1})$, where $b$ is the branching factor (we follow [3] and use $b = 3$).

We picked this particular bilingual bitext parsing formalism for two reasons. First, it is relevant to machine translation research. Several researchers have found that state-of-the-art performance can be attained using grammars that mix terminals and non-terminals in their rules [8, 14]. Second, the randomized pruning method is most competitive in cases where the dynamic program has a sufficiently high degree. We did experiments on monolingual parsing that showed that the improvements were not significant for most sentence lengths, and inferior to the coarse-to-fine method of [25].

The bitext parsing version of the randomized pruning algorithm is very similar to the monolingual case. Rather than being over constituent spans, our auxiliary variables in the bitext case are over *induced alignments* of synchronous derivations. A pair of words is *aligned* if it is emitted by the same synchronous rule. Note that this includes many-to-many and null alignments since several or zero lexical elements can be emitted by a single rule. Given two aligned sentences, the auxiliary variables $A_{i,j}$ are the $pq$ binary random variables indicating whether word $i$ is aligned with word $j$.

To compare our approximate inference procedure to exact inference, we follow previous work [15, 29] and measure the $\mathbb{L}_2$ distance between the pruned expectations and the exact expectations.[5]

## 4.1 Results

We ran our experiments on the Chinese Treebank (and its English translation) [39], limiting the product of the sentence lengths of the two sentences to $p \times q \leq 130$. This was necessary because computing exact expectations (as needed for comparing to our baseline) quickly becomes prohibitive. Note that our pruning method, in contrast, can handle much longer sentences without problem—one pass through all 1493 sentences with a product length of less than 1000 took 28 minutes on one 2.66GHz Xeon CPU.

We used the BerkeleyAligner [21] to obtain high-precision, intersected alignments to construct the high-confidence set $M'$ of auxiliary variables needed for $k^*$ (Section 3.2)—in other words, to construct the support of the selection chain $S^{(i)}$.

For randomized pruning to be efficient, we need to be able to extract a large number of samples within the time required for computing the exact expectations. Figure 4(a) shows the average time required to compute the full dynamic program and the dynamic program required to extract a single sample for varying sentence product lengths. The ratio between the two (explicitly shown in

Figure 4(b)) increases with the sentence lengths, and reaches three orders of magnitude, making it possible to average over a large number of samples, while still greatly reducing computation time.

We can compute expectations for many samples very efficiently, but how accurate are the approximated expectations? Figure 4(c) shows that averaging over several masks reduces bias significantly. In particular, the bias increases considerably for longer sentences when only a single sample is used, but remains roughly constant when we average multiple samples. To determine the number of samples in this experiment, we measured the time required for exact inference, and ran the auxiliary variable sampler for half of that time. The main point of Figure 4(c) is to show that under realistic running time conditions, the bias of the auxiliary variable sampler stays roughly constant as a function of sentence length.

Finally, we compared the auxiliary variable algorithm to tic-tac-toe pruning, a heuristic proposed in [40] and improved in [41]. Tic-tac-toe is an algorithm that efficiently precomputes a figure of merit for each bispan. This figure of merit incorporates an inside score and an outside score. To compute this score, we used a product of the two IBM model 1 scores (one for each directionality). When a bispan figure of merit falls under a threshold, it is pruned away.

In Figure 4(d), each curve corresponds to a family of heuristics with varying aggressiveness. With tic-tac-toe, aggressiveness is increased via the cut-off threshold, while with the auxiliary variable sampler, it is controlled by letting the sampler run for more iterations. For each algorithm, its coordinates correspond to the mean $\mathbb{L}_2$ bias and mean time in milliseconds per sentence. The plot shows that there is a large regime where the auxiliary variable algorithm dominates tic-tac-toe for this task. Our method is competitive up to a mean running time of about 15 sec/sentence, which is well above the typical running time one needs for realistic, large scale training.

## 5 Related work

There is a large body of related work on approximate inference techniques. When the goal is to maximize an objective function, simple beam pruning [10] can be sufficient. However, as argued in [4], beam pruning is not appropriate for computing expectations because the resulting approximation is too concentrated around the mode. To overcome this problem, [5] suggest adding a collection of samples to a beam of $k$-best estimates. Their approach is quite different to ours as no auxiliary variables are used.

Auxiliary variables are quite versatile and have been used to create MCMC algorithms that can exploit gradient information [11], efficient samplers for regression [1], for unsupervised Bayesian inference [31], automatic sampling of generic distribution [24] and non-parametric Bayesian statistics [37, 20, 36]. In computer vision, in particular, an auxiliary variable sampler developed by [30] is widely used for image segmentation [27].

## 6 Conclusion

Mask-based pruning is an effective way to speed up large dynamic programs for calculating feature expectations. Aggressive masks introduce heavy bias, while conservative ones offer only limited speed-ups. Our results show that, at least for bitext parsing, using many randomized aggressive masks generated with an auxiliary variable sampler is superior in time and bias to using a single, more conservative one. The applicability of this approach is in no way limited to the cases considered here. Randomized pruning will be most advantageous when high-order dynamic programs can be vastly sped up by masking, yet no single aggressive mask is likely to be adequate.

## Footnotes

†Work done while at the University of California at Berkeley.

[1] It may seem that Algorithm 2 is also slow, introducing a new bottleneck. However, $|s|$ is small in practice, and the constant is much smaller since it does not depend on the grammar, making this algorithm fast in practice.

[2] The second operation only needs the inside scores.

[3]There is a short and intuitive argument to justify this relaxation. Let $x^*$ be a state from $k^*$, and consider the set of paths $\mathscr{P}$ starting at $x^*$ and extended until they first return to $x^*$. Many of these paths have infinite length, however if $k^*$ is positive recurrent, $k^*(\cdot, \cdot)$, will assign probability zero to these paths. We then use the following reduction: when the chain is at $x^*$, first pick a path from $\mathscr{P}$ under the distribution induced by $k^*$ (this is a mixture of kernels). Once a path is selected, deterministically follow the edges in the path until coming back to $x^*$ (alternation of kernels). Since mixtures and alternations of $\pi$-invariant kernels preserve $\pi$-invariance, we are done.

[4]As a side note, we make the observation that this estimator is reminiscent of a structure mean field update. It is different though, since it is still an asymptotically unbiased estimator, while mean fields approximations converge in finite time to a biased estimate.

[5]More precisely, we averaged this bias across the sentence-pairs: $\text{bias}(\theta) = \frac{1}{|\mathcal{I}|} \sum_{i \in \mathcal{I}} \sum_{k=1}^{K} \Big( \mathbb{E}_{\theta,i}[f_k] - \tilde{\mathbb{E}}_{\theta,i}[f_k] \Big)^2$, where $\mathbb{E}_{\theta,i}[f]$, $\tilde{\mathbb{E}}_{\theta,i}[f]$ are shorthands notations for exact and approximate expectations.

## References

[1] J. Albert and S. Chib. Bayesian analysis of binary and polychotomous response data. *JASA*, 1993.

[2] C. Andrieu and E. Moulines. On the ergodicity properties of some adaptive MCMC algorithms. *Ann. Appl. Probab.*, 2006.

[3] P. Blunsom, T. Cohn, C. Dyer, and M. Osborne. A Gibbs sampler for phrasal synchronous grammar induction. In *EMNLP*, 2009.

[4] P. Blunsom, T. Cohn, and M. Osborne. A discriminative latent variable model for statistical machine translation. In *ACL-HLT*, 2008.

[5] P. Blunsom and M. Osborne. Probabilistic inference for machine translation. In *EMNLP*, 2008.

[6] D. Burkett and D. Klein. Two languages are better than one (for syntactic parsing). In *EMNLP '08*, 2008.

[7] E. Charniak and M. Johnson. Coarse-to-fine n-best parsing and maxent discriminative reranking. In *ACL*, 2005.

[8] D. Chiang. A hierarchical phrase-based model for statistical machine translation. In *ACL*, 2005.

[9] S. Clark and J. R. Curran. Parsing the WSJ using CCG and log-linear models. In *ACL*, 2004.

[10] M. Collins. *Head-Driven Statistical Models for Natural Language Parsing*. PhD thesis, UPenn, 1999.

[11] S. Duane, A. D. Kennedy, B. J. Pendleton, and D. Roweth. Hybrid Monte Carlo. *Physics Letters B*, 1987.

[12] J. Finkel, A. Kleeman, and C. Manning. Efficient, feature-based, conditional random field parsing. In *ACL*, 2008.

[13] J. R. Finkel, C. D. Manning, and A. Y. Ng. Solving the problem of cascading errors: Approximate Bayesian inference for linguistic annotation pipelines. In *EMNLP*, 2006.

[14] M. Galley, M. Hopkins, K. Knight, and D. Marcu. What's in a translation rule? In *HLT-NAACL*, 2004.

[15] A. Globerson and T. Jaakkola. Approximate inference using planar graph decomposition. In *NIPS*, 2006.

[16] J. Goodman. Parsing algorithms and metrics. In *ACL*, 1996.

[17] J. Goodman. Global thresholding and multiple-pass parsing. In *EMNLP*, 1997.

[18] M. Johnson. Joint and conditional estimation of tagging and parsing models. In *ACL*, 2001.

[19] M. Johnson, T. L. Griffiths, and S. Goldwater. Bayesian inference for PCFGs via Markov Chain Monte Carlo. In *ACL*, 2007.

[20] P. Liang, M. I. Jordan, and B. Taskar. A permutation-augmented sampler for Dirichlet process mixture models. In *ICML*, 2007.

[21] P. Liang, B. Taskar, and D. Klein. Alignment by agreement. In *NAACL*, 2006.

[22] D. J. C. MacKay. *Information Theory, Inference and Learning Algorithms*. Cambridge U. Press, 2003.

[23] I. W. McKeague and W. Wefelmeyer. Markov chain Monte Carlo and Rao-Blackwellization. *Statistical Planning and Inference*, 2000.

[24] R. Neal. Slice sampling. *Annals of Statistics*, 2000.

[25] S. Petrov and D. Klein. Improved inference for unlexicalized parsing. In *HLT-NAACL '07*, 2007.

[26] S. Petrov and D. Klein. Discriminative log-linear grammars with latent variables. In *NIPS*, 2008.

[27] J. Shi and J. Malik. Normalized cuts and image segmentation. *IEEE Transactions on Pattern Analysis and Machine Intelligence*, 2000.

[28] D. Smith and N. Smith. Bilingual parsing with factored estimation: Using english to parse korean. In *EMNLP '04*, 2004.

[29] D. A. Smith and J. Eisner. Dependency parsing by belief propagation. In *EMNLP*, 2008.

[30] R. H. Swendsen and J. S. Wang. Nonuniversal critical dynamics in MC simulations. *Rev. Lett*, 1987.

[31] M. A. Tanner and W. H. Wong. The calculation of posterior distributions by data augmentation. *JASA*, 1987.

[32] B. Taskar, D. Klein, M. Collins, D. Koller, and C. Manning. Max-margin parsing. In *EMNLP*, 2004.

[33] L. Tierney. Markov chains for exploring posterior distributions. *The Annals of Statistics*, 1994.

[34] I. Titov and J. Henderson. Loss minimization in parse reranking. In *EMNLP*, 2006.

[35] J. Turian, B. Wellington, and I. D. Melamed. Scalable discriminative learning for natural language parsing and translation. In *NIPS*, 2006.

[36] J. Van Gael, Y. Saatci, Y. W. Teh, and Z. Ghahramani. Beam sampling for the infinite hidden Markov model. In *ICML*, 2008.

[37] S. G. Walker. Sampling the Dirichlet mixture model with slices. *Communications in Statistics - Simulation and Computation*, 2007.

[38] J. Wolfe, A. Haghighi, and D. Klein. Fully distributed em for very large datasets. In *ICML '08*, 2008.

[39] N. Xue, F-D Chiou, and M. Palmer. Building a large-scale annotated Chinese corpus. In *COLING*, 2002.

[40] H. Zhang and D. Gildea. Stochastic lexicalized inversion transduction grammar for alignment. In *ACL*, 2005.

[41] H. Zhang, C. Quirk, R. C. Moore, and D. Gildea. Bayesian learning of non-compositional phrases with synchronous parsing. In *ACL*, 2008.

